# *Learning to Control an Unstable System with Forward Modeling*

**Michael I. Jordan**
Brain and Cognitive Sciences
MIT
Cambridge, MA 02139

**Robert A. Jacobs**
Computer and Information Sciences
University of Massachusetts
Amherst, MA 01003

## ABSTRACT

The forward modeling approach is a methodology for learning control when data is available in distal coordinate systems. We extend previous work by considering how this methodology can be applied to the optimization of quantities that are distal not only in space but also in time.

In many learning control problems, the output variables of the controller are not the natural coordinates in which to specify tasks and evaluate performance. Tasks are generally more naturally specified in "distal" coordinate systems (e.g., endpoint coordinates for manipulator motion) than in the "proximal" coordinate system of the controller (e.g., joint angles or torques). Furthermore, the relationship between proximal coordinates and distal coordinates is often not known a priori and, if known, not easily inverted.

The forward modeling approach is a methodology for learning control when training data is available in distal coordinate systems. A *forward model* is a network that learns the transformation from proximal to distal coordinates so that distal specifications can be used in training the controller (Jordan & Rumelhart, 1990). The forward model can often be learned separately from the controller because it depends only on the dynamics of the controlled system and not on the closed-loop dynamics.

In previous work, we studied forward models of kinematic transformations (Jordan, 1988, 1990) and state transitions (Jordan & Rumelhart, 1990). In the current paper,

we go beyond the spatial credit assignment problems studied in those papers and broaden the application of forward modeling to include cases of temporal credit assignment (cf. Barto, Sutton, & Anderson, 1983; Werbos, 1987). As discussed below, the function to be modeled in such cases depends on a time integral of the closed-loop dynamics. This fact has two important implications. First, the data needed for learning the forward model can no longer be obtained solely by observing the instantaneous state or output of the plant. Second, the forward model is no longer independent of the controller: If the parameters of the controller are changed by a learning algorithm, then the closed-loop dynamics change and so does the mapping from proximal to distal variables. Thus the learning of the forward model and the learning of the controller can no longer be separated into different phases.

# 1   FORWARD MODELING

In this section we briefly summarize our previous work on forward modeling (see also Nguyen & Widrow, 1989 and Werbos, 1987).

## 1.1   LEARNING A FORWARD MODEL

Given a fixed control law, the learning of a forward model is a system identification problem. Let $\mathbf{z} = g(\mathbf{s}, \mathbf{u})$ be a system to be modeled, where $\mathbf{z}$ is the output or the state-derivative, $\mathbf{s}$ is the state, and $\mathbf{u}$ is the control. We require the forward model to minimize the cost functional

$$J_m = \frac{1}{2} \int (\mathbf{z} - \hat{\mathbf{z}})^T (\mathbf{z} - \hat{\mathbf{z}}) dt. \tag{1}$$

where $\hat{\mathbf{z}} = \hat{g}(\mathbf{s}, \mathbf{u}, \mathbf{v})$ is the parameterized function computed by the model. Once the minimum is found, backpropagation through the model provides an estimate $\frac{\partial \hat{\mathbf{z}}}{\partial \mathbf{u}}$ of the system Jacobian matrix $\frac{\partial \mathbf{z}}{\partial \mathbf{u}}$ (cf. Jordan, 1988).

## 1.2   LEARNING A CONTROLLER

Once the forward model is sufficiently accurate, it can be used in the training of the controller. Backpropagation through the model provides derivatives that indicate how to change the outputs of the controller. These derivatives can be used to change the parameters of the controller by a further application of backpropagation. Figure 1 illustrates the general procedure.

This procedure minimizes the "distal" cost functional

$$J = \frac{1}{2} \int (\mathbf{z}^* - \mathbf{z})^T (\mathbf{z}^* - \mathbf{z}) dt, \tag{2}$$

where $\mathbf{z}^*$ is a reference signal. To see this, let the controller output be given as a function $\mathbf{u} = \mathbf{f}(\mathbf{s}, \mathbf{z}^*, \mathbf{w})$ of the state $\mathbf{s}^*$, the reference signal $\mathbf{z}^*$, and a parameter vector $\mathbf{w}$. Differentiating $J$ with respect to $\mathbf{w}$ yields

$$\nabla_{\mathbf{w}} J = - \int \frac{\partial \mathbf{u}}{\partial \mathbf{w}}^T \frac{\partial \mathbf{z}}{\partial \mathbf{u}}^T (\mathbf{z}^* - \mathbf{z}) dt. \tag{3}$$

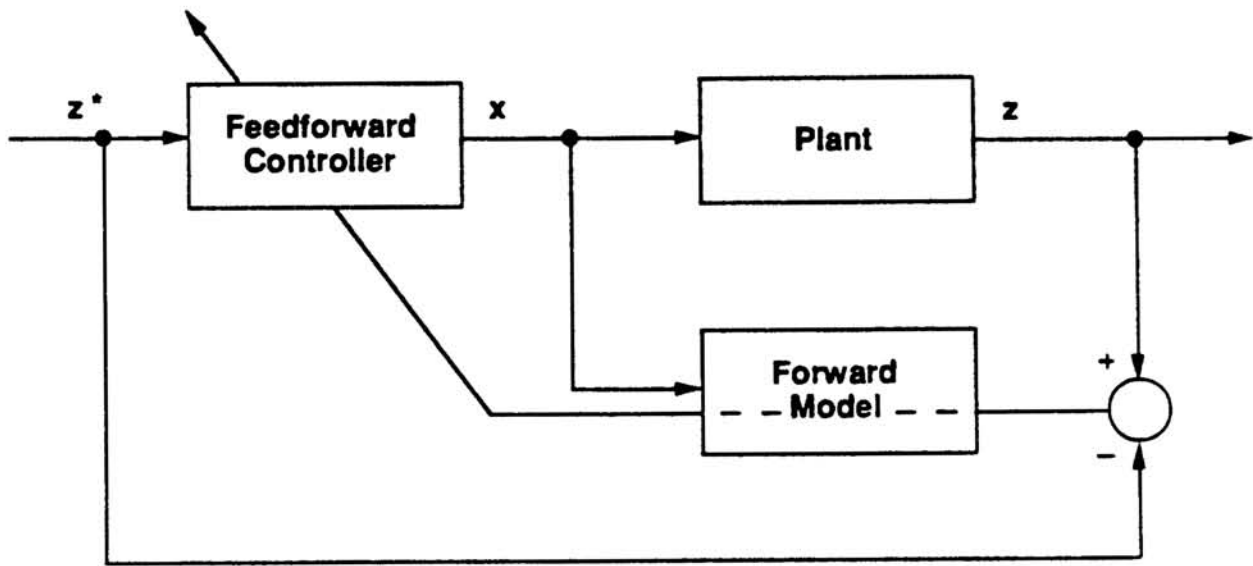

**Figure 1:** Learning a Controller. The Dashed Line Represents Backpropagation.

The Jacobian matrix $\frac{\partial \mathbf{z}}{\partial \mathbf{u}}$ cannot be assumed to be available a priori, but can be estimated by backpropagation through the forward model. Thus the error signal available for learning the controller is the estimated gradient

$$\hat{\nabla}_{\mathbf{w}} J = -\int \frac{\partial \mathbf{u}}{\partial \mathbf{w}}^T \frac{\delta \mathbf{z}}{\partial \mathbf{u}}^T (\mathbf{z}^* - \mathbf{z}) dt. \qquad (4)$$

We now consider a task in which the foregoing framework must be broadened to allow a more general form of distal task specification.

## 2   THE TASK

The task is to learn to regulate an unstable nonminimum-phase plant. We have chosen the oft-studied (e.g., Barto, Sutton, & Anderson, 1983; Widrow & Smith, 1964) problem of learning to balance an inverted pendulum on a moving cart. The plant dynamics are given by:

$$\begin{bmatrix} M + m & mlcos\theta \\ mlcos\theta & I \end{bmatrix} \begin{bmatrix} \ddot{x} \\ \ddot{\theta} \end{bmatrix} + \begin{bmatrix} -mlsin\theta \\ -mglsin\theta \end{bmatrix} \dot{\theta}^2 = \begin{bmatrix} F \\ 0 \end{bmatrix}$$

where $m$ is the mass of the pole, $M$ is the mass of the cart, $l$ is half the pole length, $I$ is the inertia of the pole around its base, and $F$ is the force applied to the cart.

The task we studied is similar to that studied by Barto, Sutton, & Anderson (1983). A state-feedback controller provides forces to the cart, and the system evolves until failure occurs (the cart reaches the end of the track or the pole reaches a critical angle). The system learns from failure; indeed, it is assumed that the *only* teaching information provided by the environment is the signal that failure has occurred.

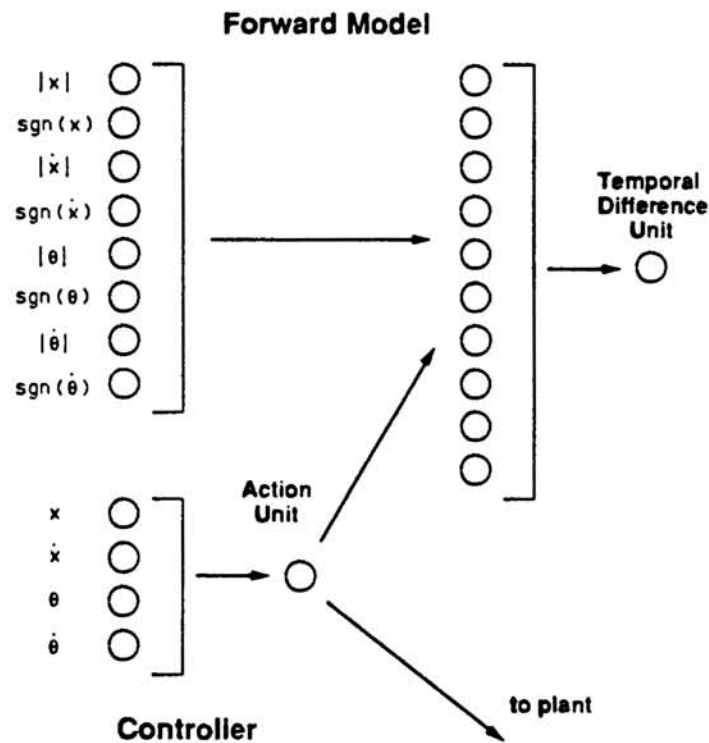

**Figure 2:** The Network Architecture

There are several differences between our task and that studied by Barto, Sutton, & Anderson (1983). First, disturbances (white noise) are provided by the environment rather than by the learning algorithm. This implies that in our experiments the level of noise seen by the controller does not diminish to zero over the course of learning. Second, we used real-valued forces rather than binary forces. Finally, we do not assume the existence of a "reset button" that reinitializes the system to the origin of state space; upon failure the system is restarted in a random configuration.

## 3   OUR APPROACH

In our approach, the control system learns a model that relates the current state of the plant and the current control signal to a prediction of future failure. We make use of a temporal difference algorithm (Sutton, 1988) to learn the transformation from (state, action) pairs to an estimate of the inverse of the time until failure. This mapping is then used as a differentiable forward model in the learning of the controller—the controller is changed so as to minimize the output of the model and thereby maximize the time until failure.

The overall system architecture is shown in Figure 2. We describe each component in detail in the following sections.

An important feature that distinguishes this architecture from previous work (e.g.,

Barto, Sutton, & Anderson, 1983) is the path from the action unit into the forward model. This path is necessary for supervised learning algorithms to be used (see also Werbos, 1987).

## 3.1    LEARNING THE FORWARD MODEL

Temporal difference algorithms learn to make long term predictions by achieving local consistency between predictions at neighboring time steps, and by grounding the chain of predictions when information from the environment is obtained. In our case, if $z(t)$ is the inverse of the time until failure, then consistency is defined by the requirement that $z^{-1}(t) = z^{-1}(t+1) + 1$. The chain is grounded by defining $z(T) = 1$, where $T$ is the time step on which failure occurs.

To learn to estimate the inverse of the time until failure, the following temporal difference error terms are used. For time steps on which failure does not occur,

$$e(t) = \frac{1}{1 + \hat{z}^{-1}(t+1)} - \hat{z}(t),$$

where $\hat{z}(t)$ denotes the output of the forward model. When failure occurs, the target for the forward model is set to unity:

$$e(t) = 1 - \hat{z}(t)$$

The error signal $e(t)$ is propagated backwards at time $t+1$ using activations saved from time $t$. Standard backpropagation is used to compute the changes to the weights.

## 3.2    LEARNING THE CONTROLLER

If the controller is performing as desired, then the output of the forward model is zero (that is, the predicted time-until-failure is infinity). This suggests that an appropriate distal error signal for the controller is zero minus the output of the forward model.

Given that the forward model has the control action as an input, the distal error can be propagated backward to the hidden units of the forward model, through the action unit, and into the controller where the weights are changed (see Figure 2).

Thus the controller is changed in such a way as to minimize the output of the forward model and thereby maximize the time until failure.

## 3.3    LEARNING THE FORWARD MODEL AND THE CONTROLLER SIMULTANEOUSLY

As the controller varies, the mapping that the forward model must learn also varies. Thus, if the forward model is to provide reasonable derivatives, it must be continuously updated as the controller changes. We find that it is possible to train the forward model and the controller simultaneously, provided that we use a larger learning rate for the forward model than for the controller.

# 4    MISCELLANY

## 4.1    RESET

Although previous studies have assumed the existence of a "reset button" that can restart the system at the origin of state space, we prefer not to make such an assumption. A reset button implies the existence of a controller that can stabilize the system, and the task of learning is to *find* such a controller. In our simulations, we restart the system at random points in state space after failure occurs.

## 4.2    REDUNDANCY

The mapping learned by the forward model depends on both the state and the action. The action, however, is itself a function of the state, so the action unit provides redundant information. This implies that the forward model could have arbitrary weights in the path from the action unit and yet make reasonable predictions. Such a model, however, would yield meaningless derivatives for learning the controller. Fortunately, backpropagation tends to produce meaningful weights for a path that is correlated with the outcome, even if that path conveys redundant information. To further bias things in our favor, we found it useful to employ a larger learning rate in the path from the action unit to the hidden units of the forward model (0.9) than in the path from the state units (0.3).

## 4.3    REPRESENTATION

As seen in Figure 2, we chose input representations that take advantage of symmetries in the dynamics of the cart-pole system. The forward model has even symmetry with respect to the state variables, whereas the controller has odd symmetry.

## 4.4    LONG-TERM BEHAVIOR

There is never a need to "turn off" the learning of the forward model. Once the pole is being successfully balanced in the presence of fluctuations, the average time until failure goes to infinity. The forward model therefore learns to predict zero in the region of state space around the origin, and the error propagated to the controller also goes to zero.

# 5    RESULTS

We ran twenty simulations starting with random initial weights. The learning rate for the controller was 0.05 and the learning rate for the forward model was 0.3, except for the connection from the action unit where the learning rate was 0.9. Eighteen runs converged to controller configurations that balanced the pole, and two runs converged on local minima. Figure 3 shows representative learning curves for six of the successful runs.

To obtain some idea of the size of the space of correct solutions, we performed an exhaustive search of a lattice in a rectangular region of weight space that contained

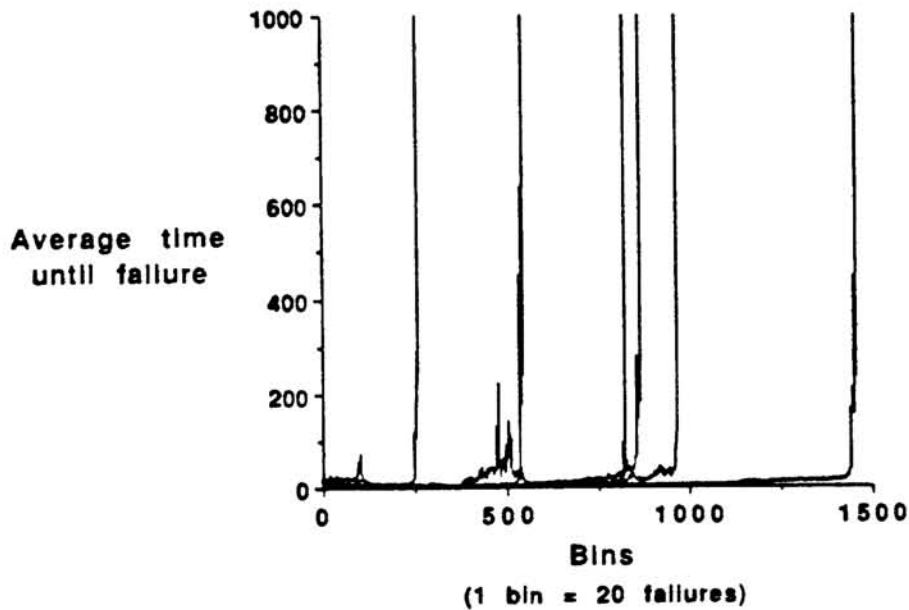

**Figure 3:** Learning Curves for Six Runs

all of the weight configurations found by our simulations. As shown in Figure 4, only 15 out of 10,000 weight configurations were able to balance the pole.

# 6    CONCLUSIONS

Previous work within the forward modeling paradigm focused on models of fixed kinematic or dynamic properties of the controlled plant (Jordan, 1988, 1990; Jordan & Rumelhart, 1990). In the current paper, the notion of a forward model is broader. The function that must be modeled depends not only on properties of the controlled plant, but also on properties of the controller. Nonetheless, the mapping is well-defined, and the results demonstrate that it can be used to provide appropriate incremental changes for the controller.

These results provide further demonstration of the applicability of supervised learning algorithms to learning control problems in which explicit target information is not available.

### Acknowledgments

The first author was supported by BRSG 2 S07 RR07047-23 awarded by the Biomedical Research Support Grant Program, Division of Research Resources, National Institutes of Health and by a grant from Siemens Corporation. The second author was supported by the Air Force Office of Scientific Research, through grant AFOSR-87-0030.

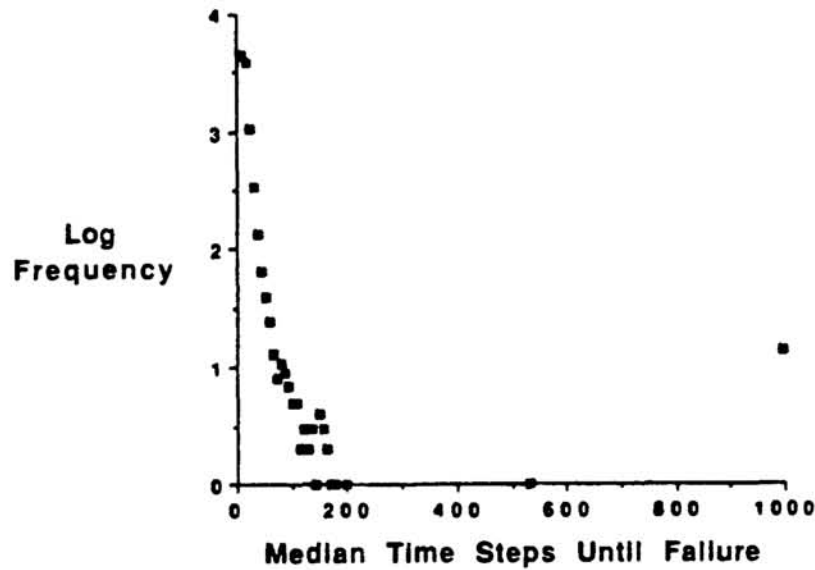

**Figure 4:** Performance of Population of Controllers

## References

Barto, A. G., Sutton, R. S., & Anderson, C. W. (1983). Neuronlike adaptive elements that can solve difficult learning control problems. *IEEE Transactions on Systems, Man, and Cybernetics, SMC-13*, 834-846.

Jordan, M. I. (1988). *Supervised learning and systems with excess degress of freedom.* (COINS Tech. Rep. 88-27). Amherst, MA: University of Massachusetts, Computer and Information Sciences.

Jordan, M. I. (1990). Motor learning and the degrees of freedom problem. In M. Jeannerod, (Ed). *Attention and Performance, XIII*. Hillsdale, NJ: Erlbaum.

Jordan, M. I. & Rumelhart, D. E. (1990). *Supervised learning with a distal teacher.* Paper in preparation.

Nguyen, D. & Widrow, B. (1989). The truck backer-upper: An example of self-learning in neural networks. In: *Proceedings of the International Joint Conference on Neural Networks*. Piscataway, NJ: IEEE Press.

Sutton, R. S. (1987). Learning to predict by the methods of temporal differences. *Machine Learning, 3*, 9-44.

Werbos, P. (1987). Building and understanding adaptive systems: A statistical/numerical approach to factory automation and brain research. *IEEE Transactions on Systems, Man, and Cybernetics, 17*, 7-20.

Widrow, B. & Smith, F. W. (1964). Pattern-recognizing control systems. In: *Computer and Information Sciences Proceedings*, Washington, D.C.: Spartan.